# A Computational Model of Eye Movements during Object Class Detection

**Wei Zhang**[†]    **Hyejin Yang**[‡*]    **Dimitris Samaras**[†]    **Gregory J. Zelinsky**[†‡]

Dept. of Computer Science[†]        Dept. of Psychology[‡]
State University of New York at Stony Brook
Stony Brook, NY 11794

{wzhang,samaras}@cs.sunysb.edu[†]    hjyang@ic.sunysb.edu[*]
Gregory.Zelinsky@stonybrook.edu[‡]

## Abstract

We present a computational model of human eye movements in an object class detection task. The model combines state-of-the-art computer vision object class detection methods (SIFT features trained using AdaBoost) with a biologically plausible model of human eye movement to produce a sequence of simulated fixations, culminating with the acquisition of a target. We validated the model by comparing its behavior to the behavior of human observers performing the identical object class detection task (looking for a teddy bear among visually complex nontarget objects). We found considerable agreement between the model and human data in multiple eye movement measures, including number of fixations, cumulative probability of fixating the target, and scanpath distance.

## 1. Introduction

Object detection is one of our most common visual operations. Whether we are driving [1], making a cup of tea [2], or looking for a tool on a workbench [3], hundreds of times each day our visual system is being asked to detect, localize, or acquire through movements of gaze objects and patterns in the world.

In the human behavioral literature, this topic has been extensively studied in the context of visual search. In a typical search task, observers are asked to indicate, usually by button press, whether a specific target is present or absent in a visual display (see [4] for a review). A primary manipulation in these studies is the number of non-target objects also appearing in the scene. A bedrock finding in this literature is that, for targets that cannot be defined by a single visual feature, target detection times increase linearly with the number of non-targets, a form of clutter or "set size" effect. Moreover, the slope of the function relating detection speed to set size is steeper (by roughly a factor of two) when the target is absent from the scene compared to when it is present. Search theorists have interpreted these findings as evidence for visual attention moving serially from one object to the next, with the human detection operation typically limited to those objects fixated by this "spotlight" of attention [5].

Object class detection has also been extensively studied in the computer vision community,

with faces and cars being the two most well researched object classes [6, 7, 8, 9]. The related but simpler task of object class recognition (target recognition without localization) has also been the focus of exciting recent work [10, 11, 12]. Both tasks use supervised learning methods to extract visual features. Scenes are typically realistic and highly cluttered, with object appearance varying greatly due to illumination, view, and scale changes. The task addressed in this paper falls between the class detection and recognition problems. Like object class detection, we will be detecting and localizing class-defined targets; unlike object class detection the test images will be composed of at most 20 objects appearing on a simple background.

Both the behavioral and computer vision literatures have strengths and weaknesses when it comes to understanding human object class detection. The behavioral literature has accumulated a great deal of knowledge regarding the conditions affecting object detection [4], but this psychology-based literature has been dominated by the use of simple visual patterns and models that cannot be easily generalized to fully realistic scenes (see [13, 14] for notable exceptions). Moreover, this literature has focused almost entirely on object-specific detection, cases in which the observer knows precisely how the target will appear in the test display (see [15] for a discussion of target non-specific search using featurally complex objects). Conversely, the computer vision literature is rich with models and methods allowing for the featural representation of object classes and the detection of these classes in visually cluttered real-world scenes, but none of these methods have been validated as models of human object class detection by comparison to actual behavioral data.

The current study draws upon the strengths of both of these literatures to produce the first joint behavioral-computational study of human object class detection. First, we use an eyetracker to quantify human behavior in terms of the number of fixations made during an object class detection task. Then we introduce a computational model that not only performs the detection task at a level comparable to that of the human observers, but also generates a sequence of simulated eye movements similar in pattern to those made by humans performing the identical detection task.

## 2. Experimental methods

An effort was made to keep the human and model experiments methodologically similar. Both experiments used training, validation (practice trials in the human experiment), and testing phases, and identical images were presented to the model and human subjects in all three of these phases. The target class consisted of 378 teddy bears scanned from [16]. Nontargets consisted of 2,975 objects selected from the Hemera Photo Objects Collection. Samples of the bear and nontarget objects are shown in Figure 1. All objects were normalized to have a bounding box area of 8,000 pixels, but were highly variable in appearance.

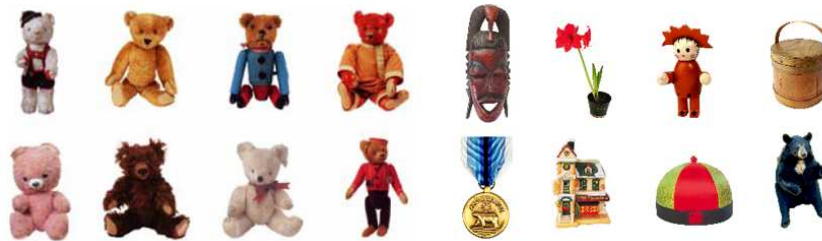

Figure 1: Representative teddy bears (left) and nontarget objects (right).

The training set consisted of 180 bears and 500 nontargets, all randomly selected. In the case of the human experiment, each of these objects was shown centered on a white background and displayed for 1 second. The testing set consisted of 180 new bears and nontar-

gets. No objects were repeated between training and testing, and no objects were repeated within either of the training or testing phases. Test images depicted 6, 13, or 20 color objects randomly positioned on a white background. A single bear was present in half (90) of these displays. Human subjects were instructed to indicate, by pressing a button, whether a teddy bear appeared among the displayed objects. Target presence and set size were randomly interleaved over trials. Each test trial in the human experiment began with the subject fixating gaze at the center of the display, and eye position was monitored throughout each trial using an eyetracker. Eight students from Stony Brook University participated in the experiment.

## 3. Model of eye movements during object class detection

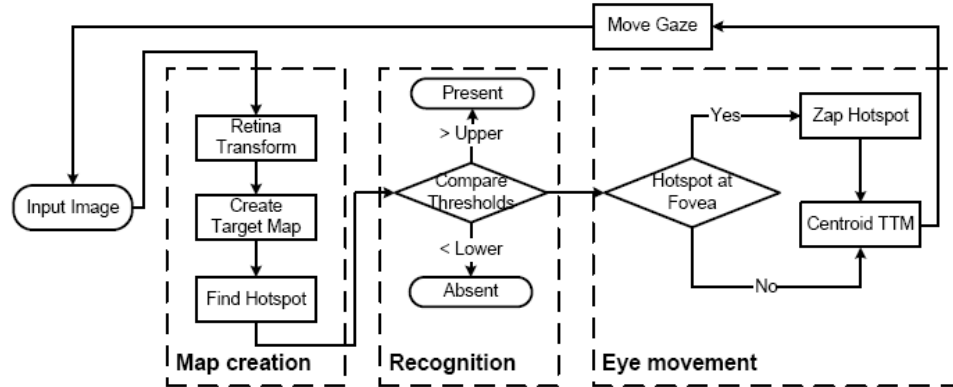

Figure 2: The flow of processing through our model.

Building on a framework described in [17, 14, 18], our model can be broadly divided into three stages (Figure 2): (1) creating a target map based on a retinally-transformed version of the input image, (2) recognizing the target using thresholds placed on the target map, and (3) the operations required in the generation of eye movements. The following sub-sections describe each of the Figure 2 steps in greater detail.

### 3.1. Retina transform
With each change in gaze position (set initially to the center of the image), our model transforms the input image so as to reflect the acuity limitations imposed by the human retina. We used the method described in [19, 20], which was shown to provide a close approximation to human acuity limitations, to implement this dynamic retina transform.

### 3.2. Create target map
Each point on the target map ranges in value between 0 and 1 and indicates the likelihood that a target is located at that point. To create the target map, we first compute interest points on the retinally-transformed image (see section 3.2.2), then compare the features surrounding these points to features of the target object class extracted during training. Two types of discriminative features were used in this study: color features and texture features.

### 3.2.1. Color features
Color has long been used as a feature for instance object recognition [21]. In our study we explore the potential use of color as a discriminative feature for an object class. Specifically, we used a normalized color histogram of pixel hues in HSV space. Because backgrounds in our images were white and therefore uninformative, we set thresholds on the saturation and brightness channels to remove these points. The hue channel was evenly divided into 11 bins and each pixel's hue value was assigned to one of these bins using binary interpolation.

Values within each bin were weighted by $1 - d$, where $d$ is the normalized unit distance to the center of the bin. The final color histogram was normalized to be a unit vector.

Given a test image, $I_t$, and its color feature, $H_t$, we compute the distances between $H_t$ and the color features of the training set $\{H_i, i = 1, ..., N\}$. The test image is labeled as: $l(I_t) = l(I_{\arg \min_{1 \le i \le N} \chi^2(H_t, H_i)})$, and the distance metric used was: $\chi^2(H_t, H_i) = \sum_{k=1}^{K} \frac{[H_t(k) - H_i(k)]^2}{H_t(k) + H_i(k)}$, where $K$ is the number of bins.

### 3.2.2. Texture features

Local texture features were extracted on the gray level images during both training and testing. To do this, we first used a Difference-of-Gaussion (DoG) operator to detect interest points in the image, then used a Scale Invariant Feature Transform (SIFT) descriptor to represent features at each of the interest point locations. SIFT features consist of a histogram representation of the gradient orientation and magnitude information within a small image patch surrounding a point [22].

AdaBoost is a feature selection method which produces a very accurate prediction rule by combining relatively inaccurate rules-of-thumb [23]. Following the method described in [11, 12], we used AdaBoost during training to select a small set of SIFT features from among all the SIFT features computed for each sample in the training set. Specifically, each training image was represented by a set of SIFT features $\{F_{i,j}, j = 1, ...n_i\}$, where $n_i$ is the number of SIFT features in sample $I_i$. To select features from this set, AdaBoost first initialized the weights of the training samples $w_i$ to $\frac{1}{2N_p}$, $\frac{1}{2N_n}$, where $N_p$ and $N_n$ are the number of positive and negative samples, respectively. For each round of AdaBoost, we then selected one feature as a weak classifier and updated the weights of the training samples. Details regarding the algorithm used for each round of boosting can be found in [12]. Eventually, $T$ features were chosen having the best ability to discriminate the target object class from the nontargets. Each of these selected features forms a weak classifier, $h_k$, consisting of three components: a feature vector, $(f_k)$, a distance threshold, $(\theta_k)$, and an output label, $(u_k)$. Only the features from the positive training samples are used as weak classifiers. For each feature vector, $F$, we compute the distance between it and the training sample, $i$, defined as $d_i = \min_{1 \le j \le n_i} D(F_{i,j}, F_0)$, then apply the classification rule:

$$h(f, \theta) = \left\{ \begin{array}{l} 1, d < \theta \\ 0, d \ge \theta \end{array} \right. . \tag{1}$$

After the desired number of weak classifiers has been found, the final strong classifier can be defined as:

$$H = \sum_{t=1}^{T} \alpha_t h_t \tag{2}$$

where $\alpha_t = log(1/\beta_t)$. Here $\beta_t = \sqrt{\frac{1 - \epsilon_t}{\epsilon_t}}$ and the classification error $\epsilon_t = \sum |u_k - l_k|$.

### 3.2.3. Validation

A validation set, consisting of the practice trials viewed by the human observers, was used to set parameters in the model. Because our model used two types of features, each having different classifiers with different outputs, some weight for combining these classifiers was needed. The validation set was used to set this weighting.

The output of the color classifier, normalized to unit length, was based on the distance $\chi^2_{min} = \min_{1 \le i \le N}$ and defined as:

$$C_{color} = \left\{ \begin{array}{l} 0, l(I_t) = 0 \\ f(\chi^2_{min}), l(I_t) = 1 \end{array} \right. \tag{3}$$

where $f(\chi^2_{min})$ is a function monotonically decreasing with respect to $\chi^2_{min}$. The strong local texture classifier, $C_{texture}$ (Equation 2), also had normalized unit output.

The weights of the two classifiers were determined based on their classification errors on the validation set:

$$W_{color} = \frac{\epsilon_t}{\epsilon_c + \epsilon_t},$$
$$W_{texture} = \frac{\epsilon_c}{\epsilon_c + \epsilon_t} \quad . \tag{4}$$

The final combined output was used to generate the values in the target map and, ultimately, to guide the model's simulated eye movements.

### 3.3. Recognition

We define the highest-valued point on the target map as the *hotspot*. Recognition is accomplished by comparing the hotspot to two thresholds, also set through validation. If the hotspot value exceeds the high target-present threshold, then the object will be recognized as an instance of the target class. If the hotspot value falls below the target-absent threshold, then the object will be classified as not belonging to the target class. Through validation, the target-present threshold was set to yield a low false positive rate and the target-absent threshold was set to yield a high true positive rate. Moreover, target-present judgments were permitted only if the hotspot was fixated by the simulated fovea. This constraint was introduced so as to avoid extremely high false positive rates stemming from the creation of false targets in the blurred periphery of the retina-transformed image.

### 3.4. Eye movement

If neither the target-present nor the target-absent thresholds are satisfied, processing passes to the eye movement stage of our model. If the simulated fovea is not on the hotspot, the model will make an eye movement to move gaze steadily toward the hotspot location. Fixation in our model is defined as the centroid of activity on the target map, a computation consistent with a neuronal population code. Eye movements are made by thresholding this map over time, pruning off values that offer the least evidence for the target. Eventually, this thresholding operation will cause the centroid of the target map to pass an eye movement threshold, resulting in a gaze shift to the new centroid location. See [18] for details regarding the eye movement generation process. If the simulated fovea does acquire the hotspot and the target-present threshold is still not met, the model will assume that a non-target was fixated and this object will be "zapped". Zapping consists of applying a negative Gaussian filter to the hotspot location, thereby preventing attention and gaze from returning to this object (see [24] for a previous computational implementation of a conceptually related operation).

## 4. Experimental results

Model and human behavior were compared on a variety of measures, including error rates, number of fixations, cumulative probability of fixating the target, and scanpath ratio (a measure of how directly gaze moved to the target). For each measure, the model and human data were in reasonable agreement.

Table 1: Error rates for model and human subjects.

|  | Total trials | Misses | | False positives | |
|---|---|---|---|---|---|
|  |  | Frequency | Rate | Frequency | Rate |
| Human | 1440 | 46 | 3.2% | 14 | 1.0% |
| Model | 180 | 7 | 3.9% | 4 | 2.2% |

Table 1 shows the error rates for the human subjects and the model, grouped by misses and false positives. Note that the data from all eight of the human subjects are shown, resulting in the greater number of total trials. There are two key patterns. First, despite the very high level of accuracy exhibited by the human subjects in this task, our model was able to

Table 2: Average number of fixations by model and human.

| Case | Target-present | | | | Target-absent | | | |
|---|---|---|---|---|---|---|---|---|
| | p6 | p13 | p20 | slope | a6 | a13 | a20 | slope |
| Human | 3.38 | 3.74 | 4.88 | 0.11 | 4.89 | 7.23 | 9.39 | 0.32 |
| Model | 2.86 | 3.69 | 5.68 | 0.20 | 3.97 | 8.30 | 10.47 | 0.46 |

achieve comparable levels of accuracy. Second, and consistent with the behavioral search literature, miss rates were larger than false positive rates for both the humans and model.

To the extent that our model offers an accurate account of human object detection behavior, it should be able to predict the average number of fixations made by human subjects in the detection task. As indicated in Table 2, this indeed is the case. Data are grouped by target-present (p), target-absent (a), and the number of objects in the scene (6, 13, 20). In all conditions, the model and human subjects made comparable numbers of fixations. Also consistent with the behavioral literature, the average number of fixations made by human subjects in our task increased with the number of objects in the scenes, and the rate of this increase was greater in the target-absent data compared to the target-present data. Both of these patterns are also present in the model data. The fact that our model is able to capture an interaction between set size and target presence in terms of the number of fixations needed for detection lends support for our method.

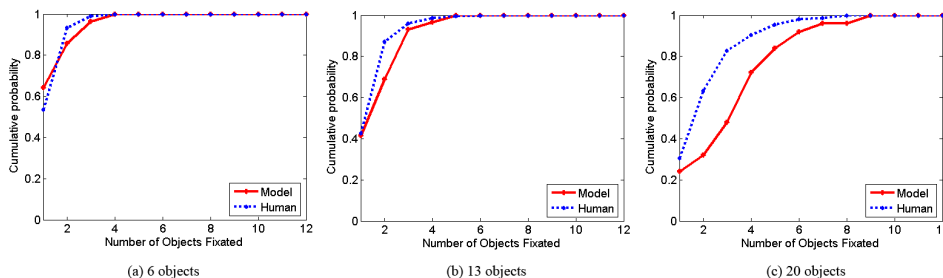

(a) 6 objects          (b) 13 objects          (c) 20 objects

Figure 3: Cumulative probability of target fixation by model and human.

Figure 3 shows the number of fixation data in more detail. Plotted are the cumulative probabilities of fixating the target as a function of the number of objects fixated during the search task. When the scene contained only 6 or 13 objects, the model and the humans fixated roughly the same number of nontargets before finally shifting gaze to the target. When the scene was more cluttered (20 objects), the model fixated an average of 1 additional nontarget relative to the human subjects, a difference likely indicating a liberal bias in our human subjects under these search conditions. Overall, these analyses suggest that our model was not only making the same number of fixations as humans, but it was also fixating the same number of nontargets during search as our human subjects.

Table 3: Comparison of model and human scanpath distance

| #Objects | 6 | 13 | 20 |
|---|---|---|---|
| Human | 1.62 | 2.20 | 2.80 |
| Model | 1.93 | 3.09 | 6.10 |
| MODEL | 1.93 | 2.80 | 3.43 |

Human gaze does not jump randomly from one item to another during search, but instead moves in a more orderly way toward the target. The ultimate test of our model would be to reproduce this orderly movement of gaze. As a first approximation, we quantify this behavior in terms of a scanpath distance. Scanpath distance is defined as the ratio of the total scanpath length (i.e., the summed distance traveled by the eye) and the distance between the target and the center of the image (i.e., the minimum distance that the eye would need to travel to fixate the target). As indicated in Table 3, the model and human data are in close agreement in the 6 and 13-object scenes, but not in the 20-object scenes. Upon closer inspection of the data, we found several cases in which the model made multiple fixations between two nontarget objects, a very unnatural behavior arising from too small of a setting for our Gaussian "zap" window. When these 6 trials were removed, the model data (MODEL) and the human data were in closer agreement.

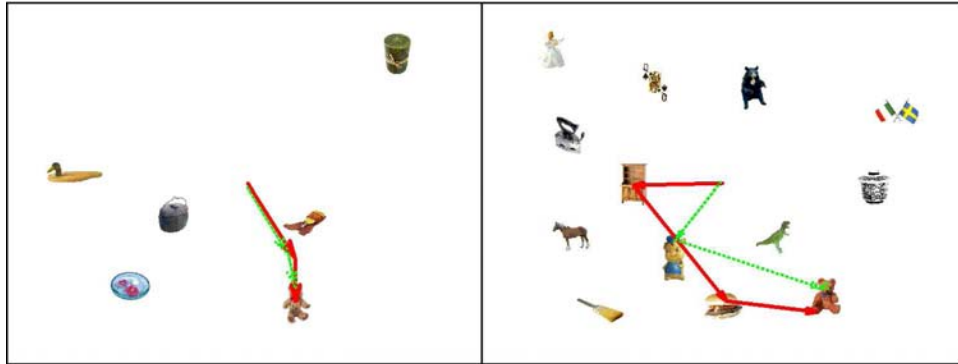

Figure 4: Representative scanpaths. Model data are shown in thick red lines, human data are shown in thin green lines.

Figure 4 shows representative scanpaths from the model and one human subject for two search scenes. Although the scanpaths do not align perfectly, there is a qualitative agreement between the human and model in the path followed by gaze to the target.

## 5. Conclusion

Search tasks do not always come with specific targets. Very often, we need to search for dogs, or chairs, or pens, without any clear idea of the visual features comprising these objects. Despite the prevalence of these tasks, the problem of object class detection has attracted surprisingly little research within the behavioral community [15], and has been applied to a relatively narrow range of objects within the computer vision literature [6, 7, 8, 9]. The current work adds to our understanding of this important topic in two key respects. First, we provide a detailed eye movement analysis of human behavior in an object class detection task. Second, we incorporate state-of-the-art computer vision object detection methods into a biologically plausible model of eye movement control, then validate this model by comparing its behavior to the behavior of our human observers. Computational models capable of describing human eye movement behavior are extremely rare [25]; the fact that the current model was able to do so for multiple eye movement measures lends strength to our approach. Moreover, our model was able to detect targets nearly as well as the human observers while maintaining a low false positive rate, a difficult standard to achieve in a generic detection model. Such agreement between human and model suggests that simple color and texture features may be used to guide human attention and eye movement in an object class detection task.

Future computational work will explore the generality of our object class detection method to tasks with visually complex backgrounds, and future human work will attempt to use

neuroimaging techniques to localize object class representations in the brain.

**Acknowledgments**

This work was supported by grants from the NIMH (R01-MH63748) and ARO (DAAD19-03-1-0039) to G.J.Z.

# References

[1] M. F. Land and D. N. Lee. Where we look when we steer. *Nature*, 369(6483):742–744, 1994.

[2] M. F. Land and M. Hayhoe. In what ways do eye movements contribute to everyday activities. *Vision Research*, 41(25-36):3559–3565, 2001.

[3] G. Zelinsky, R. Rao, M. Hayhoe, and D. Ballard. Eye movements reveal the spatio-temporal dynamics of visual search. *Psychological Science*, 8:448–453, 1997.

[4] J. Wolfe. Visual search. In *H. Pashler (Ed.), Attention*, pages 13–71. London: University College London Press, 1997.

[5] E. Weichselgartner and G. Sperling. Dynamics of automatic and controlled visual attention. *Science*, 238(4828):778–780, 1987.

[6] H. Schneiderman and T. Kanade. A statistical method for 3d object detection applied to faces and cars. In *CVPR*, volume I, pages 746–751, 2000.

[7] P. Viola and M.J. Jones. Rapid object detection using a boosted cascade of simple features. In *CVPR*, volume I, pages 511–518, 2001.

[8] S. Agarwal and D. Roth. Learning a sparse representation for object detection. In *ECCV*, volume IV, page 113, 2002.

[9] Wolf Kienzle, Gökhan H. Bakır, Matthias O. Franz, and Bernhard Schölkopf. Face detection - efficient and rank deficient. In *NIPS*, 2004.

[10] R. Fergus, P. Perona, and A. Zisserman. Object class recognition by unsupervised scale-invariant learning. In *CVPR03*, volume II, pages 264–271, 2003.

[11] A. Opelt, M. Fussenegger, A. Pinz, and P. Auer. Weak hypotheses and boosting for generic object detection and recognition. In *ECCV04*, volume II, pages 71–84, 2004.

[12] W. Zhang, B. Yu, G. Zelinsky, and D. Samaras. Object class recognition using multiple layer boosting with multiple features. In *CVPR*, 2005.

[13] L. Itti and C. Koch. A saliency-based search mechanism for overt and covert shifts of visual attention. *Vision Research*, 40:1489–1506, 2000.

[14] R. Rao, G. Zelinsky, M. Hayhoe, and D. Ballard. Eye movements in iconic visual search. *Vision Research*, 42:1447–1463, 2002.

[15] D. T. Levin, Y. Takarae, A. G. Miner, and F. Keil. Efficient visual search by category: Specifying the features that mark the difference between artifacts and animal in preattentive vision. *Perception and Psychophysics*, 63(4):676–697, 2001.

[16] P. Cockrill. *The teddy bear encyclopedia*. New York: DK Publishing, Inc., 2001.

[17] R. Rao, G. Zelinsky, M. Hayhoe, and D. Ballard. Modeling saccadic targeting in visual search. In *NIPS*, 1995.

[18] G. Zelinsky. *Itti, L., Rees, G. and Tsotos, J.(Eds.), Neurobiology of attention*, chapter Specifying the components of attention in a visual search task, pages 395–400. Elsevier, 2005.

[19] W.S. Geisler and J.S. Perry. A real-time foveated multi-resolution system for low-bandwidth video communications. In *Human Vision and Electronic Imaging, SPIE Procceddings*, volume 3299, pages 294–305, 1998.

[20] J.S. Perry and W.S. Geisler. Gaze-contingent real-time simulation of arbitrary visual fields. In *SPIE*, 2002.

[21] M.J. Swain and D.H. Ballard. Color indexing. *IJCV*, 7(1):11–32, November 1991.

[22] D.G. Lowe. Distinctive image features from scale-invariant keypoints. *IJCV*, 60(2):91–110, November 2004.

[23] Y. Freund and R.E. Schapire. A decision-theoretic generalization of on-line learning and an application to boosting. *Journal of Computer and System Sciences*, 55(1):119–139, 1997.

[24] K. Yamada and G. Cottrell. A model of scan paths applied to face recognition. In *Seventeenth Annual Cognitive Science Conference*, pages 55–60, 1995.

[25] C. M. Privitera and L. W. Stark. Algorithms for defining visual regions-of-interest: comparison with eye fixations. *PAMI*, 22:970–982, 2000.
